# Bumptrees for Efficient Function, Constraint, and Classification Learning

## Stephen M. Omohundro
International Computer Science Institute
1947 Center Street, Suite 600
Berkeley, California 94704

## Abstract

A new class of data structures called "bumptrees" is described. These structures are useful for efficiently implementing a number of neural network related operations. An empirical comparison with radial basis functions is presented on a robot arm mapping learning task. Applications to density estimation, classification, and constraint representation and learning are also outlined.

## 1 WHAT IS A BUMPTREE?

A bumptree is a new geometric data structure which is useful for efficiently learning, representing, and evaluating geometric relationships in a variety of contexts. They are a natural generalization of several hierarchical geometric data structures including oct-trees, k-d trees, balltrees and boxtrees. They are useful for many geometric learning tasks including approximating functions, constraint surfaces, classification regions, and probability densities from samples. In the function approximation case, the approach is related to radial basis function neural networks, but supports faster construction, faster access, and more flexible modification. We provide empirical data comparing bumptrees with radial basis functions in section 2.

A bumptree is used to provide efficient access to a collection of functions on a Euclidean space of interest. It is a complete binary tree in which a leaf corresponds to each function of interest. There are also functions associated with each internal node and the defining constraint is that each interior node's function must be everwhere larger than each of the

functions associated with the leaves beneath it. In many cases the leaf functions will be peaked in localized regions, which is the origin of the name. A simple kind of bump function is spherically symmetric about a center and vanishes outside of a specified ball. Figure 1 shows the structure of a two-dimensional bumptree in this setting.

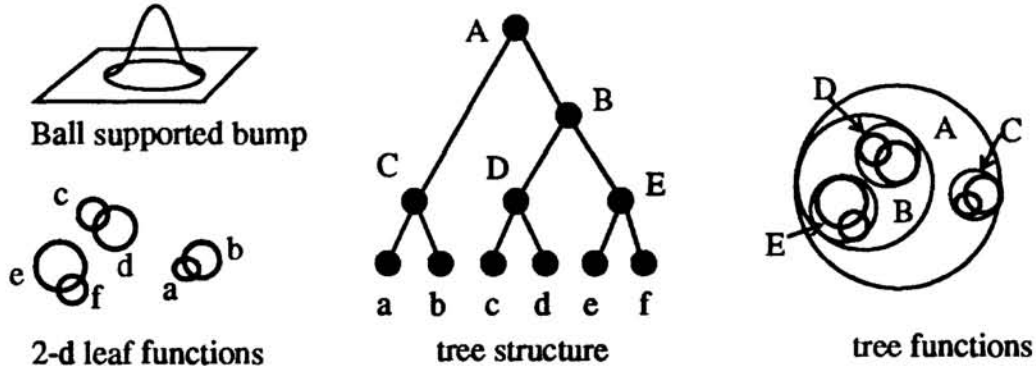

Figure 1: A two-dimensional bumptree.

A particularly important special case of bumptrees is used to access collections of Gaussian functions on multi-dimensional spaces. Such collections are used, for example, in representing smooth probability distribution functions as a Gaussian mixture and arises in many adaptive kernel estimation schemes. It is convenient to represent the quadratic exponents of the Gaussians in the tree rather than the Gaussians themselves. The simplest approach is to use quadratic functions for the internal nodes as well as the leaves as shown in Figure 2, though other classes of internal node functions can sometimes provide faster access.

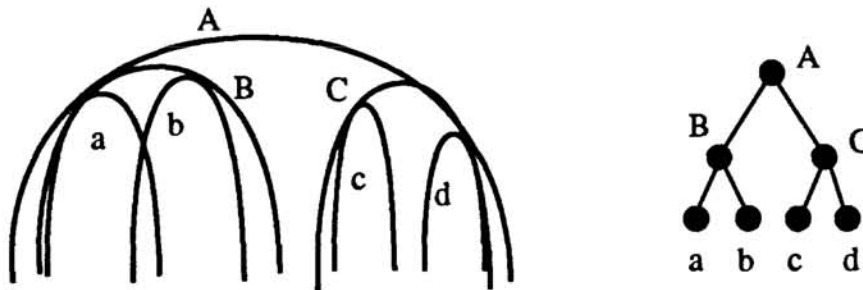

Figure 2: A bumptree for holding Gaussians.

Many of the other hierarchical geometric data structures may be seen as special cases of bumptrees by choosing appropriate internal node functions as shown in Figure 3. Regions may be represented by functions which take the value 1 inside the region and which vanish outside of it. The function shown in Figure 3D is aligned along a coordinate axis and is constant on one side of a specified value and decreases quadratically on the other side. It is represented by specifying the coordinate which is cut, the cut location, the constant value (0 in some situations), and the coefficient of quadratic decrease. Such a function may be evaluated extremely efficiently on a data point and so is useful for fast pruning operations. Such evaluations are effectively what is used in (Sproull, 1990) to implement fast nearest neighbor computation. The bumptree structure generalizes this kind of query to allow for different scales for different points and directions. The empirical results presented in the next section are based on bumptrees with this kind of internal node function.

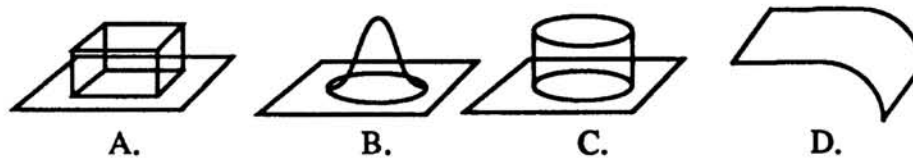

Figure 3: Internal bump functions for A) oct-trees, kd-trees, boxtrees (Omohundro, 1987), B) and C) for balltrees (Omohundro, 1989), and D) for Sproull's higher performance kd-tree (Sproull, 1990).

There are several approaches to choosing a tree structure to build over given leaf data. Each of the algorithms studied for balltree construction in (Omohundro, 1989) may be applied to the more general task of bumptree construction. The fastest approach is analogous to the basic k-d tree construction technique (Friedman, *et. al*, 1977) and is top down and recursively splits the functions into two sets of almost the same size. This is what is used in the simulations described in the next section. The slowest but most effective approach builds the tree bottom up, greedily deciding on the best pair of functions to join under a single parent node. Intermediate in speed and quality are incremental approaches which allow one to dynamically insert and delete leaf functions.

Bumptrees may be used to efficiently support many important queries. The simplest kind of query presents a point in the space and asks for all leaf functions which have a value at that point which is larger than a specified value. The bumptree allows a search from the root to prune any subtrees whose root function is smaller than the specified value at the point. More interesting queries are based on branch and bound and generalize the nearest neighbor queries that k-d trees support. A typical example in the case of a collection of Gaussians is to request all Gaussians in the set whose value at a specified point is within a specified factor (say .001) of the Gaussian whose value is largest at that point. The search proceeds down the most promising branches first, continually maintains the largest value found at any point, and prunes away subtrees which are not within the given factor of the current largest function value.

## 2  THE ROBOT MAPPING LEARNING TASK

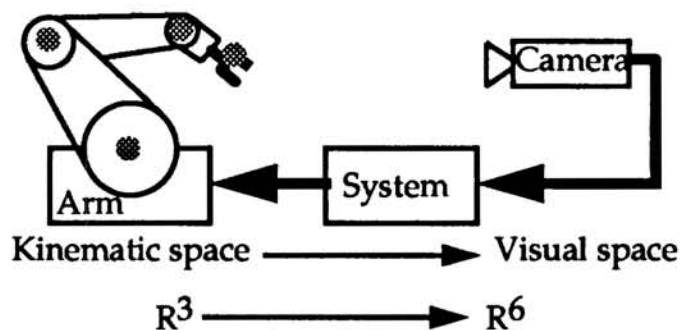

Figure 4: Robot arm mapping task.

Figure 4 shows the setup which defines the mapping learning task we used to study the effectiveness of the balltree data structure. This setup was investigated extensively by (Mel, 1990) and involves a camera looking at a robot arm. The kinematic state of the arm is defined by three angle control coordinates and the visual state by six visual coordinates of highlighted spots on the arm. The mapping from kinematic to visual space is a nonlinear map from three dimensions to six. The system attempts to learn this mapping by flailing the arm around and observing the visual state for a variety of randomly chosen kinematic states. From such a set of random input/output pairs, the system must generalize the mapping to inputs it has not seen before. This mapping task was chosen as fairly representative of typical problems arising in vision and robotics.

The radial basis function approach to mapping learning is to represent a function as a linear combination of functions which are spherically symmetric around chosen centers

$f(x) = \sum_i w_i g_i (x - x_i)$ . In the simplest form, which we use here, the basis functions are

centered on the input points. More recent variations have fewer basis functions than sample points and choose centers by clustering. The timing results given here would be in terms of the number of basis functions rather than the number of sample points for a variation of this type. Many forms for the basis functions themselves have been suggested. In our study both Gaussian and linearly increasing functions gave similar results. The coefficients of the radial basis functions are chosen so that the sum forms a least squares best fit to the data. Such fits require a time proportional to the cube of the number of parameters in general. The experiments reported here were done using the singular value decomposition to compute the best fit coefficients.

The approach to mapping learning based on bumptrees builds local models of the mapping in each region of the space using data associated with only the training samples which are nearest that region. These local models are combined in a convex way according to "influence" functions which are associated with each model. Each influence function is peaked in the region for which it is most salient. The bumptree structure organizes the local models so that only the few models which have a great influence on a query sample need to be evaluated. If the influence functions vanish outside of a compact region, then the tree is used to prune the branches which have no influence. If a model's influence merely dies off with distance, then the branch and bound technique is used to determine contributions that are greater than a specified error bound.

If a set of bump functions sum to one at each point in a region of interest, they are called a "partition of unity". We form influence bumps by dividing a set of smooth bumps (either Gaussians or smooth bumps that vanish outside a sphere) by their sum to form an easily computed partiton of unity. Our local models are affine functions determined by a least squares fit to local samples. When these are combined according to the partition of unity, the value at each point is a convex combination of the local model values. The error of the full model is therefore bounded by the errors of the local models and yet the full approximation is as smooth as the local bump functions. These results may be used to give precise bounds on the average number of samples needed to achieve a given approximation error for functions with a bounded second derivative. In this approach, linear fits are only done on a small set of local samples, avoiding the computationally expensive fits over the whole data set required by radial basis functions. This locality also allows us to easily update the model online as new data arrives.

If $b_i(x)$ are bump functions such as Gaussians, then $n_i(x) = \dfrac{b_i(x)}{\sum_j b_j(x)}$ forms a partition

of unity. If $m_i(x)$ are the local affine models, then the final smoothly interpolated approx-

imating function is $f(x) = \sum_i n_i(x)\, m_i(x)$. The influence bumps are centered on the

sample points with a width determined by the sample density. The affine model associated
with each influence bump is determined by a weighted least squares fit of the sample points
nearest the bump center in which the weight decreases with distance.

Because it performs a global fit, for a given number of samples points, the radial basis func-
tion approach achieves a smaller error than the approach based on bumptrees. In terms of
construction time to achieve a given error, however, bumptrees are the clear winner.Figure
5 shows how the mean square error for the robot arm mapping task decreases as a function
of the time to construct the mapping.

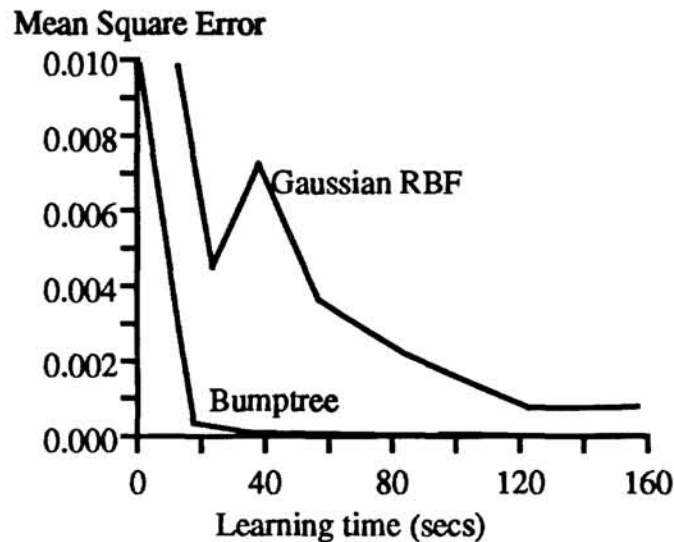

Figure 5: Mean square error as a function of learning time.

Perhaps even more important for applications than learning time is retrieval time. Retrieval
using radial basis functions requires that the value of each basis function be computed on
each query input and that these results be combined according to the best fit weight matrix.
This time increases linearly as a function of the number of basis functions in the represen-
tation. In the bumptree approach, only those influence bumps and affine models which are
not pruned away by the bumptree retrieval need perform any computation on an input. Fig-
ure 6 shows the retrieval time as a function of number of training samples for the robot map-
ping task. The retrieval time for radial basis functions crosses that for balltrees at about 100
samples and increases linearly off the graph. The balltree algorithm has a retrieval time
which empirically grows very slowly and doesn't require much more time even when
10,000 samples are represented.

While not shown here, the representation may be improved in both size and generalization
capacity by a best first merging technique. The idea is to consider merging two local models
and their influence bumps into a single model. The pair which increases the error the least

is merged first and the process is repeated until no pair is left whose meger wouldn't exceed an error criterion. This algorithm does a good job of discovering and representing linear parts of a map with a single model and putting many higher resolution models in areas with strong nonlinearities.

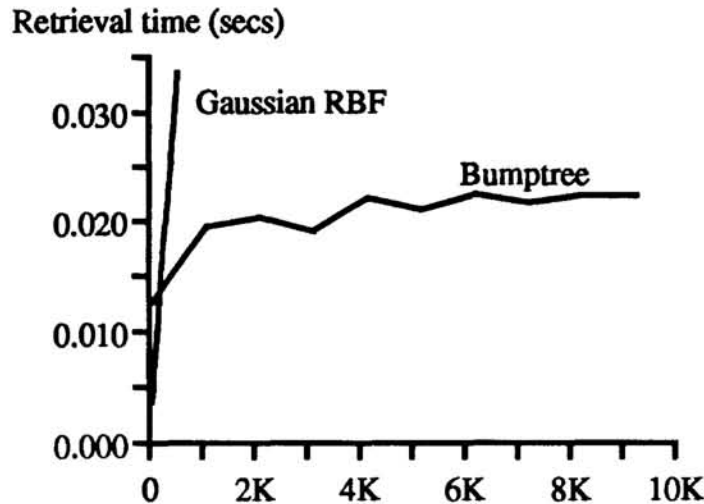

Figure 6: Retrieval time as a function of number of training samples.

## 3  EXTENSIONS TO OTHER TASKS

The bumptree structure is useful for implementing efficient versions of a variety of other geometric learning tasks (Omohundro, 1990). Perhaps the most fundamental such task is density estimation which attempts to model a probability distribution on a space on the basis of samples drawn from that distribution. One powerful technique is adaptive kernel estimation (Devroye and Gyorfi, 1985). The estimated distribution is represented as a Gaussian mixture in which a spherically symmetric Gaussian is centered on each data point and the widths are chosen according to the local density of samples. A best-first merging technique may often be used to produce mixtures consisting of many fewer non-symmetric Gaussians. A bumptree may be used to find and organize such Gaussians. Possible internal node functions include both quadratics and the faster to evaluate functions shown in Figure 3D.

It is possible to efficiently perform many operations on probability densities represented in this way. The most basic query is to return the density at a given location. The bumptree may be used with branch and bound to achieve retrieval in logarithmic expected time. It is also possible to quickly find marginal probabilities by integrating along certain dimensions. The tree is used to quickly identify the Gaussian which contribute. Conditional distributions may also be represented in this form and bumptrees may be used to compose two such distributions.

Above we discussed mapping learning and evaluation. In many situations there are not the natural input and output variables required for a mapping. If a probability distribution is peaked on a lower dimensional surface, it may be thought of as a constraint. Networks of

constraints which may be imposed in any order among variables are natural for describing many problems. Bumptrees open up several possibilities for efficiently representing and propagating smooth constraints on continuous variables. The most basic query is to specify known external constraints on certain variables and allow the network to further impose whatever constraints it can. Multi-dimensional product Gaussians can be used to represent joint ranges in a set of variables. The operation of imposing a constraint surface may be thought of as multiplying an external constraint Gaussian by the function representing the constraint distribution. Because the product of two Gaussians is a Gaussian, this operation always produces Gaussian mixtures and bumptrees may be used to facilitate the operation.

A representation of constraints which is more like that used above for mappings constructs surfaces from local affine patches weighted by influence functions. We have developed a local analog of principle components analysis which builds up surfaces from random samples drawn from them. As with the mapping structures, a best-first merging operation may be used to discover affine structure in a constraint surface.

Finally, bumptrees may be used to enhance the performance of classifiers. One approach is to directly implement Bayes classifiers using the adaptive kernel density estimator described above for each class's distribution function. A separate bumptree may be used for each class or with a more sophisticated branch and bound, a single tree may be used for the whole set of classes.

In summary, bumptrees are a natural generalization of several hierarchical geometric access structures and may be used to enhance the performance of many neural network like algorithms. While we compared radial basis functions against a different mapping learning technique, bumptrees may be used to boost the retrieval performance of radial basis functions directly when the basis functions decay away from their centers. Many other neural network approaches in which much of the network does not perform useful work for every query are also susceptible to sometimes dramatic speedups through the use of this kind of access structure.

## References

L. Devroye and L. Gyorfi. (1985) *Nonparametric Density Estimation: The L1 View*, New York: Wiley.

J. H. Friedman, J. L. Bentley and R. A. Finkel. (1977) An algorithm for finding best matches in logarithmic expected time. *ACM Trans. Math. Software* 3:209-226.

B. Mel. (1990) *Connectionist Robot Motion Planning, A Neurally-Inspired Approach to Visually-Guided Reaching*, San Diego, CA: Academic Press.

S. M. Omohundro. (1987) Efficient algorithms with neural network behavior. *Complex Systems* 1:273-347.

S. M. Omohundro. (1989) Five balltree construction algorithms. *International Computer Science Institute Technical Report* TR-89-063.

S. M. Omohundro. (1990) Geometric learning algorithms. *Physica D* 42:307-321.

R. F. Sproull. (1990) Refinements to Nearest-Neighbor Searching in k-d Trees. *Sutherland, Sproull and Associates Technical Report* SSAPP #184c, *to appear in Algorithmica*.
